# Multi-State Time Delay Neural Networks
# for Continuous Speech Recognition

**Patrick Haffner**
CNET Lannion A TSS/RCP
22301 LANNION, FRANCE
haffner@lannion.cnet.fr

**Alex Waibel**
Carnegie Mellon University
Pittsburgh, PA 15213
ahw@cs.cmu.edu

## Abstract

We present the "Multi-State Time Delay Neural Network" (MS-TDNN) as an extension of the TDNN to robust word recognition. Unlike most other hybrid methods, the MS-TDNN embeds an alignment search procedure into the connectionist architecture, and allows for word level supervision. The resulting system has the ability to manage the sequential order of subword units, while optimizing for the recognizer performance. In this paper we present extensive new evaluations of this approach over speaker-dependent and speaker-independent connected alphabet.

## 1 INTRODUCTION

Classification based Neural Networks (NN) have been successfully applied to phoneme recognition tasks. Extending those classification capabilities to word recognition is an important research direction in speech recognition. However, connectionist architectures do not model time alignment properly, and they have to be combined with a Dynamic Programming (DP) alignment procedure to be applied to word recognition. Most of these "hybrid" systems (Bourlard, 1989) take advantage of the powerful and well tried probabilistic formalism provided by Hidden Markov Models (HMM) to make use of a reliable alignment procedure. However, the use of this HMM formalism strongly limits one's choice of word models and classification procedures.

MS-TDNNs, which do not use this HMM formalism, suggest new ways to design speech recognition systems in a connectionist framework. Unlike most hybrid systems where connectionist procedures replace some parts of a pre-existing system, MS-TDNNs are designed from scratch as a global Neural Network that performs word recognition. No bootstrapping is required from an HMM, and we can apply learning procedures that correct the recognizer's errors explicitly. These networks have been successfully tested on difficult word recognition tasks, such as speaker-dependent connected alphabet recognition (Haffner et al, 1991a) and speaker-independent telephone digit recognition (Haffner and Waibel, 1991b). Section 2 presents an overview of hybrid Connectionist/HMM architectures and training procedures. Section 3 describes the MS-TDNN architecture. Section 4 presents our novel training procedure. In section 5, MS-TDNNs are tested on speaker-dependent and speaker-independent continuous alphabet recognition.

## 2    HYBRID SYSTEMS

HMMs are currently the most efficient and commonly used approach for large speech recognition tasks: their modeling capacity, however limited, fits many speech recognition problems fairly well (Lee, 1988). The main limit to the modelling capacity of HMMs is the fact that trainable parameters must be interpretable in a probabilistic framework to be reestimated using the Baum-Welch algorithm with the Maximal Likelihood Estimation training criterion (MLE).

Connectionist learning techniques used in NNs (generally error back-propagation) allow for a much wider variety of architectures and parameterization possibilities. Unlike HMMs, NNs model discrimination surfaces between classes rather than the complete input/output distributions (as in HMMs) : their parameters are only trained to minimize some error criterion. This gain in data modeling capacity, associated with a more discriminant training procedure, has permitted improved performance on a number of speech tasks, especially those in which modeling sequential information is not necessary. For instance, Time Delay Neural Networks have been applied, with high performance, to phoneme classification (Waibel et al, 1989). To extend this performance to word recognition, one has to combine a front-end NN with a procedure performing time alignment, usually based on DP. A variety of alignment procedures and training methods have been proposed for those "hybrid" systems.

### 2.1  TIME ALIGNMENT

To take into account the time distortions that may appear within its boundaries, a word is generally modeled by a sequence of states $(1,...,s,...,N)$ that can have variable durations. The score of a word in the vocabulary accumulates frame-level scores which are a function of the output $\vec{Y}(t) = (Y_1(t), ...,Y_j(t))$ of the front end NN

$$O = Max_{\{T_1, ..., T_{N+1}\}} \sum_{s=1}^{N} \left[ \sum_{t \geq T_s}^{t < T_{s+1}} Score_s(\vec{Y}(t)) \right] \qquad (1)$$

The DP algorithm finds the optimal alignment $\{T_1, ..., T_{N+1}\}$ which maximizes this word score. A variety of Score functions have been proposed for Eq.(1). They are most often treated as likelihoods, to apply the probabilistic Viterbi alignment algorithm.

### 2.1.1 NN outputs probabilities

Outputs of classification based NNs have been shown to approximate Bayes probabilities, provided that they are trained with a proper objective function (Bourlard, 1989). For instance, we can train our front-end NN to output, at each time frame, state probabilities that can be used by a Viterbi alignment procedure (to each state $s$ there corresponds a NN

output $i(s)$). Eq.(1) gives the resulting word *log (likelihood)* as a sum of frame-level *log(-likelihoods)* which are written[1]:

$$Score_s(\vec{Y}(t)) = \log(Y_{i(s)}(t)) \qquad (2)$$

### 2.1.2 Comparing NN output to a reference vector

The front end NN can be interpreted as a system remapping the input to a single density continuous HMM (Bengio, 1991). In the case of identity covariance matrices, Eq.(1) gives the *log(likelihood)* for the k-th word (after Viterbi alignment) as a sum of distances between the NN frame-level output and a reference vector associated with the current state[2].

$$Score_s(\vec{Y}(t)) = \left\| \vec{Y}(t) - \vec{Y}^s \right\|^2 \qquad (3)$$

Here, the reference vectors ($\vec{Y}^1, ..., \vec{Y}^s, ..., \vec{Y}^N$) correspond to the means of gaussian PDFs, and can be estimated with the Baum-Welch algorithm.

## 2.2 TRAINING

The first hybrid models that were proposed (Bourlard, 1989; Franzini, 1991) optimized the state-level NN (with gradient descent) and the word-level HMM (with Baum-Welch) separately. Even though each level of the system may have reached a local optimum of its cost function, training is potentially suboptimal for the given complete system. Global optimization of hybrid connectionist/HMM systems requires a unified training algorithm, which makes use of global gradient descent (Bridle, 1990).

# 3    THE MS-TDNN ARCHITECTURE

MS-TDNNs have been designed to extend TDNNs classification performance to the word level, within the simplest possible connectionist framework. Unlike the hybrid methods presented in the previous section, the HMM formalism is not taken as the underlying framework here, but many of the models developed within this formalism are applicable to MS-TDNNs.

## 3.1 FRAME-LEVEL TDNN ARCHITECTURE

All the MS-TDNNs architectures described in this paper use the front-end TDNN architecture (Waibel et al, 1989), shown in Fig.1, at the state level. Each unit of the first hidden layer receives input from a 3-frame window of input coefficients. Similarly, each unit in the second hidden layer receives input from a 5-frame window of outputs of the first hidden layer. At this level of the system (2nd hidden layer), the network produces, at each time frame, the scores for the desired phonetic features. Phoneme recognition TDNNs are trained in a time-shift invariant way by integrating over time the output of a single state.

## 3.2 BASELINE MS-TDNN

With MS-TDNNs, we have extended the formalism of TDNNs to incorporate time alignment. The front-end TDNN architecture has *I* output units, whose activations (ranging

---

1. State prior probabilities would add a constant term to Eq.(2)
2. State transition probabilities add an offset to Eq.(3)

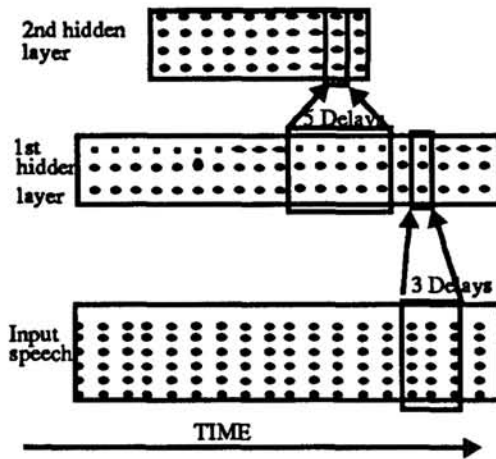

Figure 1: Frame-Level TDNN

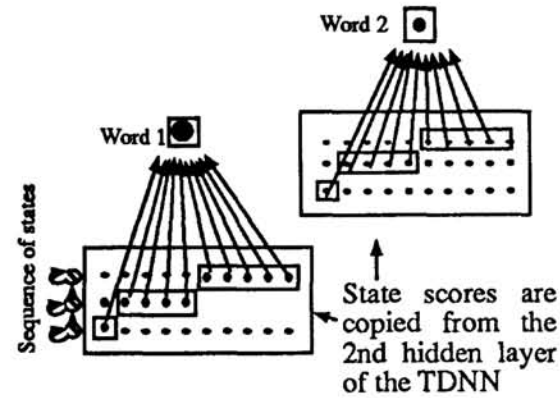

Figure 2: MS-TDNN

from 0 to 1) represent the frame-level scores. To each state s corresponds a TDNN output $i(s)$. Different states may share the same output (for instance with phone models). The DP procedure, as described in Eq.(1), determines the sequence of states producing the maximum sum of activations[3]:

$$Score_s(\vec{Y}(t)) = Y_{i(s)} \qquad (4)$$

The frame-level score used in the MS-TDNN combines the advantages of being simple with that of having a formal description as an extension of the TDNN accumulation process to multiple states. It becomes possible to model the accumulation process as a connectionist word unit that sums the activations from the best sequence of incoming state units, as shown in Fig.2. This is mostly useful during the back-propagation phase: at each time frame, we imagine a virtual connection between the active state unit and the word unit, which is used to backpropagate the error at the word level down to the state level.[4]

### 3.3 EXTENDING MS-TDNNs

In the previous section, we presented the baseline MS-TDNN architecture. We now present extensions to the word-level architecture, which provide additional trainable parameters. Eq.(4) is extended as:

$$Score_s(\vec{Y}(t)) = Weight_i \cdot Y_{i(s)} + Bias_i \qquad (5)$$

---

3. This equation is not very different from Eq.(2) presented in the previous section, however, all attempts to use $log(Y_i(t))$ instead of $Y_i(t)$ have resulted in unstable learning runs, that have never converged properly. During the test phase, the two approaches may be functionally not very different. Outputs that affect the error rate in a critical way are mostly those of the correct word and the best incorrect word, especially when they are close. We have observed that frame level scores which play a key role in discrimination are close to 1.0: the two scores become asymptotically equivalent (less 1): $log(Y_i(t)) \sim Y_i(t) - 1$.

4. The alignment path found by the DP routine during the forward phase is "frozen", so that it can be represented as a connectionist accumulation unit during the backward phase. The problem is that, after modification of the weights, this alignment path may no longer be the optimal one. Practical consequences of this seem minimal.

*Weight*$_i$ allows to weight differently the importance of each state belonging to the same word. We do not have to assume that each part of a speech pattern contains an equal amount of information.

*Bias*$_i$ is analog to a transition log(probability) in a HMM.

However, we have observed that a small variation in the value of those parameters may alter recognition performance a lot. The choice of a proper training procedure is critical. Our gradient back-propagation algorithm has been selected for its efficient training of the parameters of the front-end TDNN ; as our training procedure is global, we have also applied it to train *Weight*$_i$ and *Bias*$_i$, but with some difficulty.

In section 4.1, we show that they are useful to shift the word scores so that a sigmoid function separates the correct words (output *1*) properly from the incorrect ones (output *0*).

## 3.4  SEQUENCE MODELS

We design very simple state sequence models by hand that may use phonetic knowledge (phone models) or may not (word models).

**Phone Models:** The phonetic representation of a word is transcribed as a sequence of states. As an example shown in Fig.3, the letter 'p' combines 3 phone units. P captures the closure and the burst of this stop consonant. P-IY is a co-articulation unit. The phone IY is recognized in a context independent way. This phone is shared with all the other e-set letters. States are duplicated to enforce minimal phone durations.

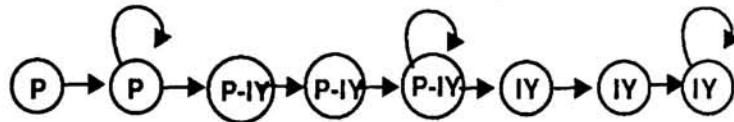

Figure 3 Phone Model for 'p'

**Word Models:** No specific phonemic meaning is associated with the states of a word. Those states cannot be shared with other words.

**Transition States:** One can add specialized transition units that are trained to detect this transition more explicitly : the resulting stabilization in segmentation yields an increase in performance. This method is however sensitive to a good bootstrapping of our system on proper phone boundaries, and has so far only been applied to speaker dependent alphabet recognition.

# 4   TRAINING

In many speech recognition systems, a large discrepancy is found between the training procedure and the testing procedure. The training criterion, generally Maximum Likelihood Estimation, is very far from the word accuracy the system is expected to maximize. Good performance depends on a large quantity of data, and on proper modeling. With MS-TDNNs, we suggest optimization procedures which explicitly attempt to minimize the number of word substitutions ; this approach represents a move towards systems in which the training objective is maximum word accuracy. The same global gradient back-propagation is applied to the whole system, from the output word units down to the input units. Each desired word is associated with a segment of speech with known boundaries, and this association represents a learning sample. The DP alignment procedure is applied between the known word boundaries. We describe now three training procedures we have applied to MS-TDNNs.

## 4.1 STANDARD BACK-PROPAGATION WITH SIGMOIDAL OUTPUTS

Word outputs $Q_k = f(W_k \cdot O_k + B_k)$ are compared to word targets ($1$ for the desired word, $0$ for the other words), and the resulting error is back-propagated. $O_k$ is the DP sum given by Eq.(1) for the k-th word in the vocabulary, $f$ is the sigmoid function, $W_k$ gives the slope of the sigmoid and $B_k$ is a bias term, as shown in Fig.4. They are trained so that the sigmoid function separates the correct word (Output $1$) form the incorrect words (Output $0$) properly. When the network is trained with the additional parameters of Eq.(5), $Weight_i$ and $Bias_i$ can account for these sigmoid slope and bias.

MS-TDNNs are applied to word recognition problems where classes are highly confusable. The score of the best incorrect word may be very close to the score of the correct word: in this case, the slope and the bias are parameters which are difficult to tune, and the learning procedure has problems to attain the 0 and 1 target values. To overcome those difficulties, we have developed new training techniques which do not require the use of a sigmoid function and of fixed word targets.

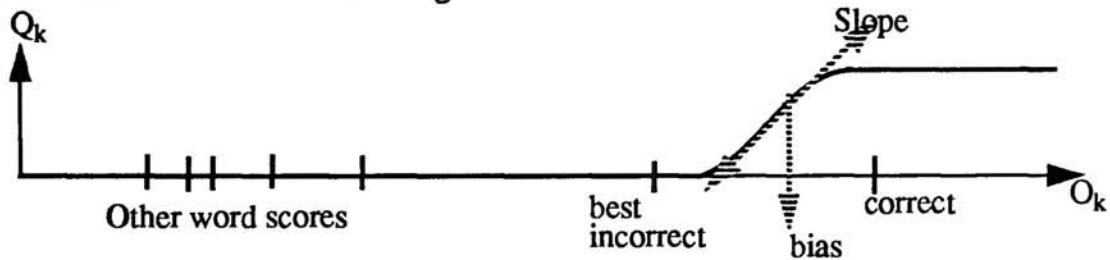

Fig.4. The sigmoid Function

## 4.2 ON-LINE CORRECTION OF CLASSIFICATION ERRORS

The testing procedure recognizes the word (or the string of words) with the largest output, and there is an error when this is not the correct word. As the goal of the training procedure is to minimize the number of errors, the "ideal" procedure would be, each time a classification error has occurred, to observe where it comes from, and to modify the parameters of the system so that it no longer happens.

The MS-TDNN has to recognize the correct word *CoWo*. There is a *training error* if, for an incorrect word *InWo*, one has $O_{InWo} > O_{CoWo} - m$. No sigmoid function is needed to compare these outputs, $m$ is an additional margin to ensure the robustness of the training procedure. *Only* in the event of a training error do we modify the parameters of the MS-TDNN. The word targets are moving (for instance, the target score for an incorrect word is $O_{CoWo} - m$) instead of fixed ($0$ or $1$).

This technique overcomes the difficulties due to the use of an output sigmoid function. Moreover, the number of incorrect words whose output is actually modified is greatly reduced: this is very helpful in training under-represented classes, as the numbers of positive and negative examples become much more balanced.

Compared to the more traditional training technique (with a sigmoid) presented in the previous section, large increases in training speed and word accuracy were observed.

## 4.3 FUZZY WORD BOUNDARIES

Training procedures we have presented so far do not take into account the fact that the sample words may come from continuous speech. The main difficulty is that their straightforward extension to continuous speech would not be computationally feasible, as the set

of possible training classes will consist of all the possible strings of words.We have adopted a staged approach: we modify the training procedure, so that it matches the continuous recognition conditions more and more closely, while remaining computationally feasible.

The first step deals with the problem of word boundaries. During training, known word boundaries give additional information that the system uses to help recognition. But this information is not available when testing. To overcome this problem when learning a correct word (noted *CoWo*), we take as the correct training token the triplet *PreWo-CoWo-NexWo* (*PreWo* is the preceding correct word, *NexWo* is the next correct word in the sentence). All the other triplets *PreWo-InWo-NexWo* are considered as incorrect. These triplets are aligned between the beginning known boundary of *PrevWo* and the ending known boundary of *NexWo*. What is important is that no precise boundary information is given for *CoWo*.

The word classification training criterion presented here only minimizes word substitutions. In connected speech, one has to deal with deletions and insertions errors: procedures to describe them as classification errors are currently being developed.

## 5  EXPERIMENTS ON CONNECTED ALPHABET

Recognizing spoken letters is considered one of the most challenging small-vocabulary tasks in speech recognition. The vocabulary, consisting of the 26 letters of the American English alphabet, is highly confusable, especially among subsets like the E-set ('B','C','D','E','G','P','T','V','Z') or ('M','N'). In all experiments, as input parameters, 16 filterbank melscale spectrum coefficients are computed at a 10msec frame rate. Phone models are used.

### 5.1  SPEAKER DEPENDENT ALPHABET

Our database consists of 1000 connected strings of letters, some corresponding to grammatical words and proper names, others simply random. There is an average of five letters per string. The learning procedure is described in §4.1 and applied to the extended MS-TDNN (§3.3), with a bootstrapping phase where phone labels are used to give the alignment of the desired word. During testing, time alignment is performed over the whole sentence. A one-stage DP algorithm (Ney, 1984) for connected words (with no grammar) is used in place of the isolated word DP algorithm used in the training phase. The additional use of minimal word durations, word entrance penalties and word boundary detectors has reduced the number of word insertions and deletions (in the DP algorithm) to an acceptable level. On two speakers, the word error rates are respectively 2.4% and 10.3%. By comparison, SPHINX, achieved error rates of 6% and 21.7%, respectively, when context-independent (as in our MS-TDNN) phone models were used. Using context-dependent models (as described in Lee, 1988), SPHINX performance achieves 4% and 16.1% error rates, respectively. No comparable results yet exist for the MS-TDNN for this case.

### 5.2  SPEAKER INDEPENDENT ALPHABET (RMspell)

Our database, a part of the DARPA Resource Management database (RM), consists of 120 speakers, spelling about 15 words each. 109 speakers (about 10,000 spelled letters) are used for training. 11 speakers (about 1000 spelled letters) are used for testing. 57 phone units, in the second hidden layer, account for the phonemes and the co-articulation units. We apply the training algorithms described in §4.2 and §4.3 to our baseline MS-

TDNN architecture (§3.2), without any additional procedure (for instance, no phonetic bootstrapping). An important difference from the experimental conditions described in the previous section is that we have kept training and testing conditions exactly similar (for instance, the same knowledge of the boundaries is used during training and testing).

Table 1: Alphabet classification errors (we do not allow for insertions or deletions errors).

| Algorithm | %Error |
|---|---|
| Known Word Boundaries (§4.2) | 5.7% |
| Fuzzy Word Boundaries (§4.3) | 6.5% |

## 6  SUMMARY

We presented in this paper MS-TDNNs, which extend TDNNs classification performance to the sequence level. They integrate the DP alignment procedure within a straightforward connectionist framework. We developed training procedures which are computationally reasonable and train the MS-TDNN in a global way. Their only supervision is the minimization of the recognizer's error rate. Experiments were conducted on speaker independent continuous alphabet recognition. The word error rates are 5.7% with known word boundaries and 6.5% with fuzzy word boundaries.

### Acknowledgments

The authors would like to express their gratitude to Denis Jouvet and Michael Witbrock, for their help writing this paper, and to Cindy Wood, for gathering the databases. This work was partially performed at CNET laboratories, and at Carnegie Mellon University, under DARPA support.

### References

Bengio, Y "Artificial Neural Networks and their Application to Sequence Recognition" Ph.D. Thesis, McGill University, Montreal, June 1991.

Bourlard, H and Morgan, N. "Merging Multilayer Perceptrons and Hidden Markov Models: Some Experiments in Continuous Speech Recognition", TR-89-033, ICSI, Berkeley, CA, July 1989

Bridle, J.S. "Alphanets: a recurrent 'neural' network architecture with a hidden Markov model interpretation." Speech Communication, "Neurospeech" issue, Feb 1990.

Franzini, M.A., Lee, K.F., and Waibel, A.H.,"Connectionist Viterbi Training: A New Hybrid Method for Continuous Speech Recognition," ICASSP, Albuquerque 1990

Haffner, P., Franzini.M. and Waibel A., " Integrating Time Alignment and Neural Networks for High Performance Continuous Speech Recognition " ICASSP, Toronto 1991a.

Haffner, P and Waibel A. "Time-Delay Neural Networks Embedding Time Alignment: a Performance Analysis" Europseech'91, Genova, September 1991b.

Lee, K.F. "Large-Vocabulary Speaker-Independent Continuous Speech Recognition: the SPHINX system", PhD Thesis, Carnegie Mellon University, 1988.

Ney, H. "The Use of a One-Stage Dynamic Programming Algorithm for Connected Word Recognition" in IEEE Trans. on Acoustics, Speech and Signal Processing. April 1984.

Waibel, A.H., Hanazawa, T., Hinton, G., Shikano, K., and Lang, K., " Phoneme Recognition using Time-Delay Neural Networks " in IEEE Transactions on Acoustics, Speech and Signal Processing 37(3):328-339, 1989.